# Stable Fitted Reinforcement Learning

**Geoffrey J. Gordon**
Computer Science Department
Carnegie Mellon University
Pittsburgh PA 15213
ggordon@cs.cmu.edu

## Abstract

We describe the reinforcement learning problem, motivate algorithms which seek an approximation to the $Q$ function, and present new convergence results for two such algorithms.

## 1  INTRODUCTION AND BACKGROUND

Imagine an agent acting in some environment. At time $t$, the environment is in some state $x_t$ chosen from a finite set of states. The agent perceives $x_t$, and is allowed to choose an action $a_t$ from some finite set of actions. The environment then changes state, so that at time $(t + 1)$ it is in a new state $x_{t+1}$ chosen from a probability distribution which depends only on $x_t$ and $a_t$. Meanwhile, the agent experiences a real-valued cost $c_t$, chosen from a distribution which also depends only on $x_t$ and $a_t$ and which has finite mean and variance.

Such an environment is called a Markov decision process, or MDP. The reinforcement learning problem is to control an MDP to minimize the expected discounted cost $\sum_t \gamma^t c_t$ for some discount factor $\gamma \in [0, 1]$. Define the function $Q$ so that $Q(x, a)$ is the cost for being in state $x$ at time 0, choosing action $a$, and behaving optimally from then on. If we can discover $Q$, we have solved the problem: at each step, we may simply choose $a_t$ to minimize $Q(x_t, a_t)$. For more information about MDPs, see (Watkins, 1989, Bertsekas and Tsitsiklis, 1989).

We may distinguish two classes of problems, online and offline. In the offline problem, we have a full model of the MDP: given a state and an action, we can describe the distributions of the cost and the next state. We will be concerned with the online problem, in which our knowledge of the MDP is limited to what we can discover by interacting with it. To solve an online problem, we may approximate the transition and cost functions, then proceed as for an offline problem (the indirect approach); or we may try to learn the $Q$ function without the intermediate step (the direct approach). Either approach may work better for any given problem: the

direct approach may not extract as much information from each observation, but the indirect approach may introduce additional errors with its extra approximation step. We will be concerned here only with direct algorithms.

Watkins' (1989) $Q$-learning algorithm can find the $Q$ function for small MDPs, either online or offline. Convergence with probability 1 in the online case was proven in (Jaakkola *et al.*, 1994, Tsitsiklis, 1994). For large MDPs, exact $Q$-learning is too expensive: representing the $Q$ function requires too much space. To overcome this difficulty, we may look for an inexpensive approximation to the $Q$ function. In the offline case, several algorithms for this purpose have been proven to converge (Gordon, 1995a, Tsitsiklis and Van Roy, 1994, Baird, 1995). For the online case, there are many fewer provably convergent algorithms. As Baird (1995) points out, we cannot even rely on gradient descent for large, stochastic problems, since we must observe two independent transitions from a given state before we can compute an unbiased estimate of the gradient. One of the algorithms in (Tsitsiklis and Van Roy, 1994), which uses state aggregation to approximate the $Q$ function, can be modified to apply to online problems; the resulting algorithm, unlike $Q$-learning, must make repeated small updates to its control policy, interleaved with comparatively lengthy periods of evaluation of the changes. After submitting this paper, we were advised of the paper (Singh *et al.*, 1995), which contains a different algorithm for solving online MDPs. In addition, our newer paper (Gordon, 1995b) proves results for a larger class of approximators.

There are several algorithms which can handle restricted versions of the online problem. In the case of a Markov chain (an MDP where only one action is available at any time step), Sutton's TD($\lambda$) has been proven to converge for arbitrary linear approximators (Sutton, 1988, Dayan, 1992). For decision processes with linear transition functions and quadratic cost functions (the so-called linear quadratic regulation problem), the algorithm of (Bradtke, 1993) is guaranteed to converge. In practice, researchers have had mixed success with approximate reinforcement learning (Tesauro, 1990, Boyan and Moore, 1995, Singh and Sutton, 1996).

The remainder of the paper is divided into four sections. In section 2, we summarize convergence results for offline $Q$-learning, and prove some contraction properties which will be useful later. Section 3 extends the convergence results to online algorithms based on TD(0) and simple function approximators. Section 4 treats nondiscounted problems, and section 5 wraps up.

## 2   OFFLINE DISCOUNTED PROBLEMS

Standard offline $Q$-learning begins with an MDP $M$ and an initial $Q$ function $q^{(0)}$. Its goal is to learn $q^{(n)}$, a good approximation to the optimal $Q$ function for $M$. To accomplish this goal, it performs the series of updates $q^{(i+1)} = T_M(q^{(i)})$, where the component of $T_M(q^{(i)})$ corresponding to state $x$ and action $a$ is defined to be

$$[T_M(q^{(i)})]_{xa} \equiv c_{xa} + \gamma \sum_y P_{xay} \min_b q^{(i)}_{yb}$$

Here $c_{xa}$ is the expected cost of performing action $a$ in state $x$; $P_{xay}$ is the probability that action $a$ from state $x$ will lead to state $y$; and $\gamma$ is the discount factor.

Offline $Q$-learning converges for discounted MDPs because $T_M$ is a contraction in max norm. That is, for all vectors $q$ and $r$,

$$\| T_M(q) - T_M(r) \| \le \gamma \| q - r \|$$

where $\| q \| \equiv \max_{x,a} | q_{xa} |$. Therefore, by the contraction mapping theorem, $T_M$ has a unique fixed point $q^*$, and the sequence $q^{(i)}$ converges linearly to $q^*$.

It is worth noting that a weighted version of offline $Q$-learning is also guaranteed to converge. Consider the iteration

$$q^{(i+1)} = (I + \alpha D(T_M - I))(q^{(i)})$$

where $\alpha$ is a positive learning rate and $D$ is an arbitrary fixed nonsingular diagonal matrix of weights. In this iteration, we update some $Q$ values more rapidly than others, as might occur if for instance we visited some states more frequently than others. (We will come back to this possibility later.) This weighted iteration is a max norm contraction, for sufficiently small $\alpha$: take two $Q$ functions $q$ and $r$, with $\| q - r \| = l$. Suppose $\alpha$ is small enough that the largest element of $\alpha D$ is $B < 1$, and let $b > 0$ be the smallest diagonal element of $\alpha D$. Consider any state $x$ and action $a$, and write $d_{xa}$ for the corresponding element of $\alpha D$. We then have

$$
\begin{aligned}
{[(I - \alpha D)q - (I - \alpha D)r]_{xa}} &\leq (1 - d_{xa})l \\
[T_M q - T_M r]_{xa} &\leq \gamma l \\
[\alpha D T_M q - \alpha D T_M r]_{xa} &\leq d_{xa}\gamma l \\
[(I - \alpha D + \alpha D T_M)q - (I - \alpha D + \alpha D T_M)r]_{xa} &\leq (1 - d_{xa})l + d_{xa}\gamma l \\
&\leq (1 - b(1 - \gamma))l
\end{aligned}
$$

so $(I - \alpha D + \alpha D T_M)$ is a max norm contraction with factor $(1 - b(1 - \gamma))$. The fixed point of weighted $Q$-learning is the same as the fixed point of unweighted $Q$-learning: $T_M(q^*) = q^*$ is equivalent to $\alpha D(T_M - I)q^* = 0$.

The difficulty with standard (weighted or unweighted) $Q$-learning is that, for MDPs with many states, it may be completely infeasible to compute $T_M(q)$ for even one value of $q$. One way to avoid this difficulty is fitted $Q$-learning: if we can find some function $M_A$ so that $M_A \circ T_M$ is much cheaper to compute than $T_M$, we can perform the fitted iteration $q^{(i+1)} = M_A(T_M(q^{(i)}))$ instead of the standard offline $Q$-learning iteration. The mapping $M_A$ implements a function approximation scheme (see (Gordon, 1995a)); we assume that $q^{(0)}$ can be represented as $M_A(q)$ for some $q$. The fitted offline $Q$-learning iteration is guaranteed to converge to a unique fixed point if $M_A$ is a nonexpansion in max norm, and to have bounded error if $M_A(q^*)$ is near $q^*$ (Gordon, 1995a).

Finally, we can define a fitted weighted $Q$-learning iteration:

$$q^{(i+1)} = (I + \alpha M_A D(T_M - I))(q^{(i)})$$

If $M_A$ is a max norm nonexpansion and $M_A^2 = M_A$ (these conditions are satisfied, for example, by state aggregation), then fitted weighted $Q$-learning is guaranteed to converge:

$$
\begin{aligned}
(I + \alpha M_A D(T_M - I))q &= ((I - M_A) + M_A(I + \alpha D(T_M - I)))q \\
&= M_A(I + \alpha D(T_M - I)))q
\end{aligned}
$$

since $M_A q = q$ for $q$ in the range of $M_A$. (Note that $q^{(i+1)}$ is guaranteed to be in the range of $M_A$ if $q^{(i)}$ is.) The last line is the composition of a max norm nonexpansion with a max norm contraction, and so is a max norm contraction.

The fixed point of fitted weighted $Q$-learning is not necessarily the same as the fixed point of fitted $Q$-learning, unless $M_A$ can represent $q^*$ exactly. However, if $M_A$ is linear, we have that

$$(I + \alpha M_A D(T_M - I))(q + c) = c + M_A(I + \alpha D(T_M - I)))(q + c)$$

for any $q$ in the range of $M_A$ and $c$ perpendicular to the range of $M_A$. In particular, if we take $c$ so that $q^* - c$ is in the range of $M_A$, and let $q = M_A q$ be a fixed point

of the weighted fitted iteration, then we have

$$
\begin{aligned}
\| q^* - q \| &= \| (I + \alpha M_A D(T_M - I))q^* - (I + \alpha M_A D(T_M - I))q \| \\
&= \| c + M_A(I + \alpha D(T_M - I))q^* - M_A(I + \alpha D(T_M - I))q \| \\
&\leq \| c \| + (1 - b(1 - \gamma)) \| q^* - q \| \\
\| q^* - q \| &\leq \frac{\| c \|}{b(1 - \gamma)}
\end{aligned}
$$

That is, if $M_A$ is linear in addition to the conditions for convergence, we can bound the error for fitted weighted $Q$-learning.

For offline problems, the weighted version of fitted $Q$-learning is not as useful as the unweighted version: it involves about the same amount of work per iteration, the contraction factor may not be as good, the error bound may not be as tight, and it requires $M_A^2 = M_A$ in addition to the conditions for convergence of the unweighted iteration. On the other hand, as we shall see in the next section, the weighted algorithm can be applied to online problems.

## 3  ONLINE DISCOUNTED PROBLEMS

Consider the following algorithm, which is a natural generalization of TD(0) (Sutton, 1988) to Markov decision problems. (This algorithm has been called "sarsa" (Singh and Sutton, 1996).) Start with some initial $Q$ function $q^{(0)}$. Repeat the following steps for $i$ from 0 onwards. Let $\pi^{(i)}$ be a policy chosen according to some predetermined tradeoff between exploration and exploitation for the $Q$ function $q^{(i)}$. Now, put the agent in $M$'s start state and allow it to follow the policy $\pi^{(i)}$ for a random number of steps $L^{(i)}$. If at step $t$ of the resulting trajectory the agent moves from the state $x_t$ under action $a_t$ with cost $c_t$ to a state $y_t$ for which the action $b_t$ appears optimal, compute the estimated Bellman error

$$
e_t = (c_t + \gamma[q^{(i)}]_{y_t b_t}) - [q^{(i)}]_{x_t a_t}
$$

After observing the entire trajectory, define $e^{(i)}$ to be the vector whose $xa$-th component is the sum of $e_t$ for all $t$ such that $x_t = x$ and $a_t = a$. Then compute the next weight vector according to the TD(0)-like update rule with learning rate $\alpha^{(i)}$

$$
q^{(i+1)} = q^{(i)} + \alpha^{(i)} M_A e^{(i)}
$$

See (Gordon, 1995b) for a comment on the types of mappings $M_A$ which are appropriate for online algorithms.

We will assume that $L^{(i)}$ has the same distribution for all $i$ and is independent of all other events related to the $i$-th and subsequent trajectories, and that $E(L^{(i)})$ is bounded. Define $d_{xa}^{(i)}$ to be the expected number of times the agent visited state $x$ and chose action $a$ during the $i$-th trajectory, given $\pi^{(i)}$. We will assume that the policies are such that $d_{xa}^{(i)} > \epsilon$ for some positive $\epsilon$ and for all $i$, $x$, and $a$. Let $D^{(i)}$ be the diagonal matrix with elements $d_{xa}^{(i)}$. With this notation, we can write the expected update for the sarsa algorithm in matrix form:

$$
E(q^{(i+1)} \,|\, q^{(i)}) = (I + \alpha^{(i)} M_A D^{(i)} (T_M - I))q^{(i)}
$$

With the exception of the fact that $D^{(i)}$ changes from iteration to iteration, this equation looks very similar to the offline weighted fitted $Q$-learning update. However, the sarsa algorithm is not guaranteed to converge even in the benign case

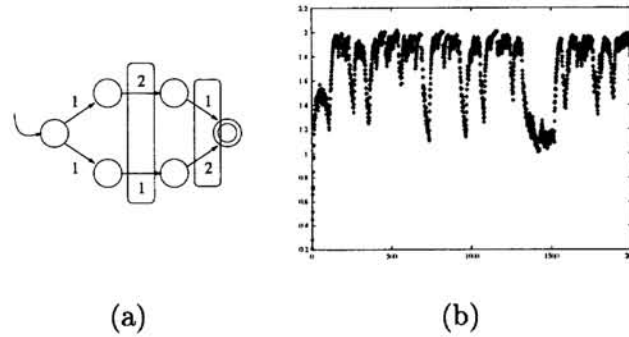

(a)                                        (b)

Figure 1: A counterexample to sarsa. (a) An MDP: from the start state, the agent may choose the upper or the lower path, but from then on its decisions are forced. Next to each arc is its expected cost; the actual costs are randomized on each step. Boxed pairs of arcs are aggregated, so that the agent must learn identical $Q$ values for arcs in the same box. We used a discount $\gamma = .9$ and a learning rate $\alpha = .1$. To ensure sufficient exploration, the agent chose an apparently suboptimal action 10% of the time. (Any other parameters would have resulted in similar behavior. In particular, annealing $\alpha$ to zero wouldn't have helped.) (b) The learned $Q$ value for the right-hand box during the first 2000 steps.

where the $Q$-function is approximated by state aggregation: when we apply sarsa to the MDP in figure 1, one of the learned $Q$ values oscillates forever. This problem happens because the frequency-of-update matrix $D^{(i)}$ can change discontinuously when the $Q$ function fluctuates slightly: when, by luck, the upper path through the MDP appears better, the cost-1 arc into the goal will be followed more often and the learned $Q$ value will decrease, while when the lower path appears better the cost-2 arc will be weighted more heavily and the $Q$ value will increase. Since the two arcs out of the initial state always have the same expected backed-up $Q$ value (because the states they lead to are constrained to have the same value), each path will appear better infinitely often and the oscillation will continue forever.

On the other hand, if we can represent the optimal $Q$ function $q^*$, then no matter what $D^{(i)}$ is, the expected sarsa update has its fixed point at $q^*$. Since the smallest diagonal element of $D^{(i)}$ is bounded away from zero and the largest is bounded above, we can choose an $\alpha$ and a $\gamma' < 1$ so that $(I + \alpha M_A D^{(i)}(T_M - I))$ is a contraction with fixed point $q^*$ and factor $\gamma'$ for all $i$. Now if we let the learning rates satisfy $\sum_i \alpha^{(i)} = \infty$ and $\sum_i (\alpha^{(i)})^2 < \infty$, convergence w.p.1 to $q^*$ is guaranteed by a theorem of (Jaakkola et al., 1994). (See also the theorem in (Tsitsiklis, 1994).)

More generally, if $M_A$ is linear and can represent $q^* - c$ for some vector $c$, we can bound the error between $q^*$ and the fixed point of the expected sarsa update on iteration $i$: if we choose an $\alpha$ and a $\gamma' < 1$ as in the previous paragraph,

$$\| E(q^{(i+1)} \,|\, q^{(i)}) - q^* \| \leq \gamma' \| q^{(i)} - q^* \| + 2\| c \|$$

for all $i$. A minor modification of the theorem of (Jaakkola et al., 1994) shows that the distance from $q^{(i)}$ to the region

$$\left\{ q \,\middle|\, \| q - q^* \| \leq 2\| c \| \frac{1}{1 - \gamma'} \right\}$$

converges w.p.1 to zero. That is, while the sequence $q^{(i)}$ may not converge, the worst it will do is oscillate in a region around $q^*$ whose size is determined by how

accurately we can represent $q^*$ and how frequently we visit the least frequent (state, action) pair.

Finally, if we follow a fixed exploration policy on every trajectory, the matrix $D^{(i)}$ will be the same for every $i$; in this case, because of the contraction property proved in the previous section, convergence w.p.1 for appropriate learning rates is guaranteed again by the theorem of (Jaakkola *et al.*, 1994).

# 4    NONDISCOUNTED PROBLEMS

When $M$ is not discounted, the $Q$-learning backup operator $T_M$ is no longer a max norm contraction. Instead, as long as every policy guarantees absorption w.p.1 into some set of cost-free terminal states, $T_M$ is a contraction in some weighted max norm. The proofs of the previous sections still go through, if we substitute this weighted max norm for the unweighted one in every case. In addition, the random variables $L^{(i)}$ which determine when each trial ends may be set to the first step $t$ so that $x_t$ is terminal, since this and all subsequent steps will have Bellman errors of zero. This choice of $L^{(i)}$ is not independent of the $i$-th trial, but it does have a finite mean and it does result in a constant $D^{(i)}$.

# 5    DISCUSSION

We have proven new convergence theorems for two online fitted reinforcement learning algorithms based on Watkins' (1989) $Q$-learning algorithm. These algorithms, sarsa and sarsa with a fixed exploration policy, allow the use of function approximators whose mappings $M_A$ are max norm nonexpansions and satisfy $M_A^2 = M_A$. The prototypical example of such a function approximator is state aggregation. For similar results on a larger class of approximators, see (Gordon, 1995b).

### Acknowledgements

This material is based on work supported under a National Science Foundation Graduate Research Fellowship and by ARPA grant number F33615-93-1-1330. Any opinions, findings, conclusions, or recommendations expressed in this publication are those of the author and do not necessarily reflect the views of the National Science Foundation, ARPA, or the United States government.

# References

L. Baird. Residual algorithms: Reinforcement learning with function approximation. In *Machine Learning (proceedings of the twelfth international conference)*, San Francisco, CA, 1995. Morgan Kaufmann.

D. P. Bertsekas and J. N. Tsitsiklis. *Parallel and Distributed Computation: Numerical Methods*. Prentice Hall, 1989.

J. A. Boyan and A. W. Moore. Generalization in reinforcement learning: safely approximating the value function. In G. Tesauro and D. Touretzky, editors, *Advances in Neural Information Processing Systems*, volume 7. Morgan Kaufmann, 1995.

S. J. Bradtke. Reinforcement learning applied to linear quadratic regulation. In S. J. Hanson, J. D. Cowan, and C. L. Giles, editors, *Advances in Neural Information Processing Systems*, volume 5. Morgan Kaufmann, 1993.

P. Dayan. The convergence of TD($\lambda$) for general lambda. *Machine Learning*, 8(3–4):341–362, 1992.

G. J. Gordon. Stable function approximation in dynamic programming. In *Machine Learning (proceedings of the twelfth international conference)*, San Francisco, CA, 1995. Morgan Kaufmann.

G. J. Gordon. Online fitted reinforcement learning. In J. A. Boyan, A. W. Moore, and R. S. Sutton, editors, *Proceedings of the Workshop on Value Function Approximation*, 1995. Proceedings are available as tech report CMU-CS-95-206.

T. Jaakkola, M. I. Jordan, and S. P. Singh. On the convergence of stochastic iterative dynamic programming algorithms. *Neural Computation*, 6(6):1185–1201, 1994.

S. P. Singh, T. Jaakkola, and M. I. Jordan. Reinforcement learning with soft state aggregation. In G. Tesauro and D. Touretzky, editors, *Advances in Neural Information Processing Systems*, volume 7. Morgan Kaufmann, 1995.

S. P. Singh and R. S. Sutton. Reinforcement learning with replacing eligibility traces. *Machine Learning*, 1996.

R. S. Sutton. Learning to predict by the methods of temporal differences. *Machine Learning*, 3(1):9–44, 1988.

G. Tesauro. Neurogammon: a neural network backgammon program. In *IJCNN Proceedings III*, pages 33–39, 1990.

J. N. Tsitsiklis and B. Van Roy. Feature-based methods for large-scale dynamic programming. Technical Report P-2277, Laboratory for Information and Decision Systems, 1994.

J. N. Tsitsiklis. Asynchronous stochastic approximation and Q-learning. *Machine Learning*, 16(3):185–202, 1994.

C. J. C. H. Watkins. *Learning from Delayed Rewards*. PhD thesis, King's College, Cambridge, England, 1989.
